# Orientation-Selective aVLSI Spiking Neurons

**Shih-Chii Liu, Jörg Kramer, Giacomo Indiveri,**
**Tobias Delbrück, and Rodney Douglas**
Institute of Neuroinformatics
University of Zurich and ETH Zurich
Winterthurerstrasse 190
CH-8057 Zurich, Switzerland

## Abstract

We describe a programmable multi-chip VLSI neuronal system that can be used for exploring spike-based information processing models. The system consists of a silicon retina, a PIC microcontroller, and a transceiver chip whose integrate-and-fire neurons are connected in a soft winner-take-all architecture. The circuit on this multi-neuron chip approximates a cortical microcircuit. The neurons can be configured for different computational properties by the virtual connections of a selected set of pixels on the silicon retina. The virtual wiring between the different chips is effected by an event-driven communication protocol that uses asynchronous digital pulses, similar to spikes in a neuronal system. We used the multi-chip spike-based system to synthesize orientation-tuned neurons using both a feedforward model and a feedback model. The performance of our analog hardware spiking model matched the experimental observations and digital simulations of continuous-valued neurons. The multi-chip VLSI system has advantages over computer neuronal models in that it is real-time, and the computational time does not scale with the size of the neuronal network.

## 1 Introduction

The sheer number of cortical neurons and the vast connectivity within the cortex are difficult to duplicate in either hardware or software. Simulations of a network consisting of thousands of neurons with a connectivity that is representative of cortical neurons can take minutes to hours on a fast Pentium, particularly if spiking behavior is simulated. The simulation time of the network increases as the size of the network increases. We have taken initial steps in mitigating the simulation time of neuronal networks by developing a multi-chip VLSI system that can support spike-based cortical processing models. The connectivity between neurons on different chips and between neurons on the same chip are reconfigurable. The receptive fields are effected by appropriate mapping of the spikes from source neurons to target neurons. A significant advantage of these hardware simulation systems is their real-time property; the simulation time of these systems does not increase with the size of the network.

In this work, we show how we synthesized orientation-tuned spiking neurons using the multi-chip system in Figure 1. The virtual connection from a selected set of neurons on

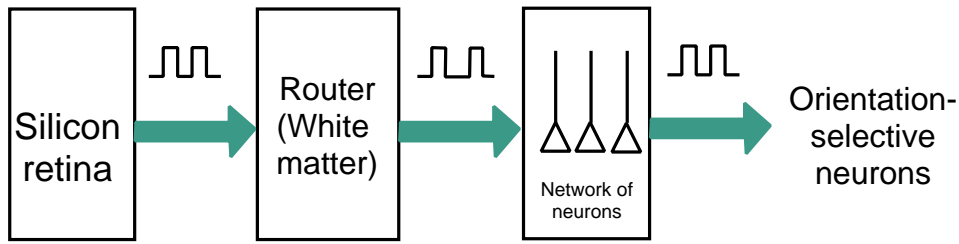

Figure 1: Block diagram of a neuromorphic multi-chip system in which virtual connections from a set of neurons on a silicon retina onto another set of neurons on a transceiver chip are effected by a microcontroller. The retina communicates through the AER protocol to the PIC when it has an active pixel. The PIC communicates with the multi-neuron chip if the retina address falls into one of its stored templates. The address from the PIC is decoded by the multi-neuron transceiver. The address of the active neuron on this array can also be communicated off-chip to another receiver/transceiver.

the retina to the target neurons on the multi-neuron transceiver chip is achieved with a PIC microcontroller and an asynchronous event-driven communication protocol. The circuit on this multi-neuron chip approximates a cortical microcircuit (Douglas and Martin, 1991).

We explored different models that have been proposed for the generation of orientation tuning in neurons of the V1 cortical area. There have been earlier attempts to use multi-chip systems for creating orientation-selective neurons (Boahen et al., 1997; Whatley et al., 1997). In the present work, the receptive fields are created in a manner similar to that described in (Whatley et al., 1997). However we extend their work and quantify the tuning curves of different models. Visual cortical neurons receive inputs from the lateral geniculate nucleus (LGN) neurons which are not orientation-selective. Models for the emergence of orientation-selectivity in cortical neurons can be divided into two groups; feedforward models and feedback models. In a feedforward model, the orientation selectivity of a cortical neuron is conferred by the spatial alignment of the LGN neurons that are presynaptic to the cortical neuron (Hubel and Wiesel, 1962). In a feedback model, a weak orientation bias provided by the LGN input is sharpened by the intracortical excitatory and/or inhibitory feedback (Somers et al., 1995; Ben-Yishai et al., 1995; Douglas et al., 1995). In this work, we quantify the tuning curves of neurons created using a feedforward model and a feedback model with global inhibition.

## 2   System Architecture

The multi-chip system (Figure 1) in this work consists of a $16 \times 16$ silicon ON/OFF retina, a PIC microcontroller, and a transceiver chip with a ring of 16 integrate-and-fire neurons and a global inhibitory neuron. All three modules communicate using the address event representation (AER) protocol (Lazzaro et al., 1993; Boahen, 1996). The communication channel signals consist of the address bits, the REQ signal, and the ACK signal. The PIC and the multi-neuron chip are both transceivers: They can both receive events and send events (Liu et al., 2001).

The retina with an on-chip arbiter can only send events. Each pixel is composed of an adaptive photoreceptor that has a rectifying temporal differentiator (Kramer, 2001) in its feedback loop as shown in Figure 2. Positive temporal irradiance transients (dark-to-bright or ON transitions) and negative irradiance transients (bright-to-dark or OFF transitions) appear at two different outputs of the pixel. The outputs are then coded in the form of asynchronous binary pulses by two neurons within the pixel. These asynchronous pulses

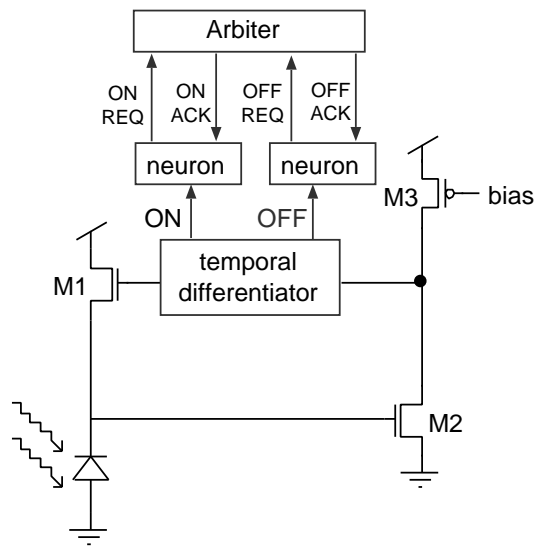

Figure 2: Pixel of the transient imager. The circuit contains a photodiode with a transistor M1 in a a source-follower configuration with a high-gain inverting amplifier (M2, M3) in a negative feedback loop. A rectifying temporal differentiator in the feedback loop extracts transient ON and OFF signals. These signals go to individual neurons that generate the REQ signals to the arbiter. In this schematic, we only show the REQ and ACK signals to the X-arbiter. The duration of the ACK signal from the X-arbiter is extended within the pixel by a global refractory bias. This duration sets the refractory period of the neuron.

are the request signals to the AER communication interface. A global parameter sets the minimum time (or refractory period) between subsequent pulses from the same output. Hence, the pixel can respond either with one pulse or multiple pulses to a transient. The pixels are arranged on a rectangular grid. The position of a pixel is encoded with a 4-bit column address (X address) and a 4-bit row address (Y address) as shown in Figure 3. An active neuron makes a request to the on-chip arbiter. If the neuron is selected by the arbiter, then the X and Y addresses which code the location of this neuron are placed on the output address bus of the chip. The retina then handshakes with the PIC microcontroller.

The multi-neuron chip has an on-chip address decoder for the incoming events and an on-chip arbiter to send events. The X address to the chip codes the identity of the neuron and the Y address codes the input synapse used to stimulate the neuron. Each neuron can be stimulated externally through an excitatory synapse or an inhibitory synapse. The excitatory neurons of this array are mutually connected via hard-wired excitatory synapses. These excitatory neurons also excite a global inhibitory neuron which in turn inhibits all the excitatory neurons. The membrane potentials of the neurons can be monitored by an on-chip scanner and the output spikes of the neurons can be monitored by the chip's AER output. The address on the output bus codes the active neuron. In this work, the excitatory neurons on the multi-neuron chip model the orientation tuning properties of simple cells in the visual cortex and the global inhibitory neuron models an inhibitory interneuron in the visual cortex.

The receptive fields of the neurons are created by configuring the connections from a subset of the source pixels on the retina onto the appropriate target neurons on the multi-neuron transceiver chip through a PIC 16C74 microcontroller. The subsets of retina pixels are determined by user-supplied templates. The microcontroller filters each retinal event to

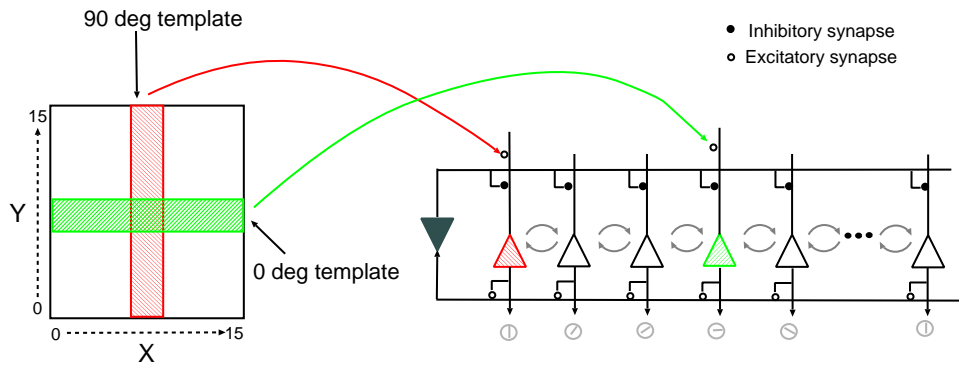

Figure 3: Spikes from a selected set of neurons within the two rectangular regions on the retina are mapped by the PIC onto the corresponding orientation-selective neurons on the transceiver chip. The light-shaded triangles mark the somas of the excitatory neurons and the dark-shaded triangle marks the soma of the global inhibitory neuron. Only two neurons, which are mapped for orthogonal orientations, were used in this experiment.

decide if it lies in one or more of the receptive fields (RFs) of the neurons on the receiver. If it does, an event is transmitted to the appropriate receiver neuron. The typical transmission time from a spike from the sender to the receiver is about 15 $\mu$s. This cycle time can be reduced by using a faster processor in place of the PIC. The retina and transceiver chips can handle handshaking cycle times on the order of 100 ns.

## 3  Neuron Circuit

The circuit of a neuron and an excitatory synapse on the transceiver chip is shown in Figure 4. The synapse circuit (M1−M4) in the left box of the figure was originally described in (Boahen, 1996). The presynaptic spike drives the transistor M4, which acts like a switch. The bias voltages $V_w$ and $V_e$ set the the strength and the dynamics of the synapse.

The circuit in the right box of Figure 4 implements a linear threshold integrate and fire neuron with an adjustable voltage threshold, spike pulse width and refractory period. The synaptic current $I_{inj}$ charges up the capacitance of the membrane $C_m$. When the membrane potential $V_{mem}$ exceeds a threshold voltage $V_{thr}$, the output of the transconductance amplifier M5−M9 switches to a voltage close to $V_{dd}$. The output of the two inverters (M10−M12 and M13−M15), $V_{out}$, also switches to $V_{dd}$. The bias voltage, $V_{pb}$, limits the current through the transconductance amplifier and the first inverter. The capacitors $C_{fb}$ and $C_m$ implement a capacitive divider that provides positive feedback to $V_{mem}$. This feedback speeds up the circuit's response and provides hysteresis to ensure that small fluctuations of $V_{mem}$ around $V_{thr}$ do not make $V_{out}$ switch erratically.

When $V_{out}$ is high, $C_m$ is discharged through transistors M20 and M21 at a rate that is dependent on $V_{pw}$. This bias voltage controls the spike's pulse width. Once $V_{mem}$ is below $V_{thr}$, the transconductance amplifier switches to ground. The first inverter then switches to $V_{dd}$ but $V_{out}$ does not immediately go to zero; it decreases linearly at a rate set by $V_{rfr}$. In this way, transistor M21 is kept on, even after $V_{mem}$ has decreased below $V_{thr}$. As long as the gate voltage of M21 is sufficiently high, the neuron is in its refractory period. Once transistor M21 is turned off, a new spike is generated in a time that is inversely proportional to the magnitude of $I_{inj}$. The spike output of the circuit is taken from the output of the first inverter.

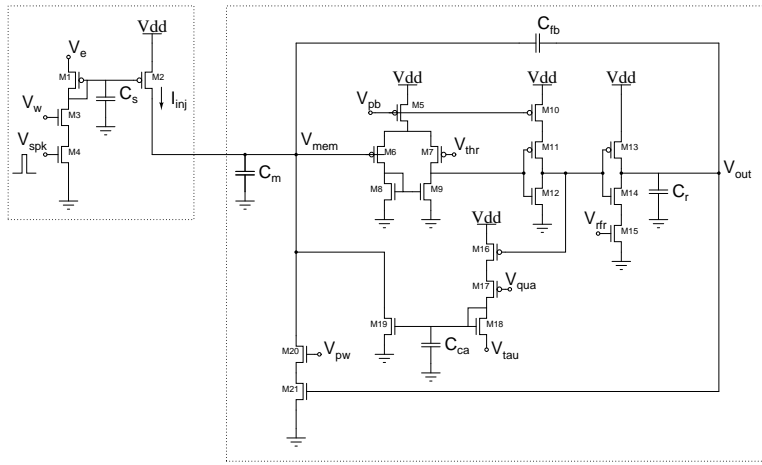

Figure 4: Circuit diagram of an excitatory synapse (left box) connected to a linear threshold integrate-and-fire neuron (right box).

Transistors M16–M19 implement a *spike frequency adaptation* mechanism (Boahen, 1996). A fixed amount of charge (set by $V_{\text{qua}}$) is dumped onto the capacitor $C_{\text{ca}}$ with every output spike. The resulting charge on $C_{\text{ca}}$ sets the current that is subtracted from the input current, and the neuron's output frequency decreases accordingly. The voltages $V_{\text{qua}}$ and $V_{\text{tau}}$ are used to set the gain and dynamics of the integrator.

## 4    System Responses

A rotating drum with a black and white strip was placed in front of the retina. The spike addresses and spike times generated by the retina and the multi-neuron chip at an image speed of 7.9 mm/s (or 89 pixels/s) of the rotating stimulus were recorded using a logic analyzer. The orientations of the stimuli are defined in Figure 3. Each pixel of the retina responded with only one spike to the transition of an edge of the stimulus because the refractory period of the pixel was set to 500 $\mu$s. The spike addresses during the time of travel of the OFF edge of a 0 deg oriented stimulus through the entire array (Figure 5(a)) indicates that almost all the pixels along a row transmitted their addresses sequentially as the edge passed by. This sequential ordering can be seen because the stimulus was oriented slightly different from 0 deg. If the stimulus was perfectly at 0 deg, then there would be a random ordering of the pixel addresses within each row. The same observation can be made for the OFF-transient spikes recorded in response to a 90 deg oriented stimulus (Figure 5(b)).

The receptive fields of two orientation-selective neurons were synthesized by mapping the OFF transient outputs of a selected set of pixels on the retina as shown in Figure 3. These two neurons have orthogonal preferred orientations. The local excitatory coupling between the neurons was disabled. There is no self excitation to each neuron so we explored only a feedforward model and a feedback model using global inhibition. We varied the size and aspect ratio of the receptive fields of the neurons by changing the template size used in the mapping of the retina spikes to the transceiver chip. The template size and aspect ratio determine the orientation responses of the neurons. The orientation response of these neurons also depends on the time constant of the neuron. On this multi-neuron chip, we do not have an explicit transistor that allows us to control the time constant. Instead, we generated a leak current through M19 in Figure 4 by controlling the source voltage of M18,

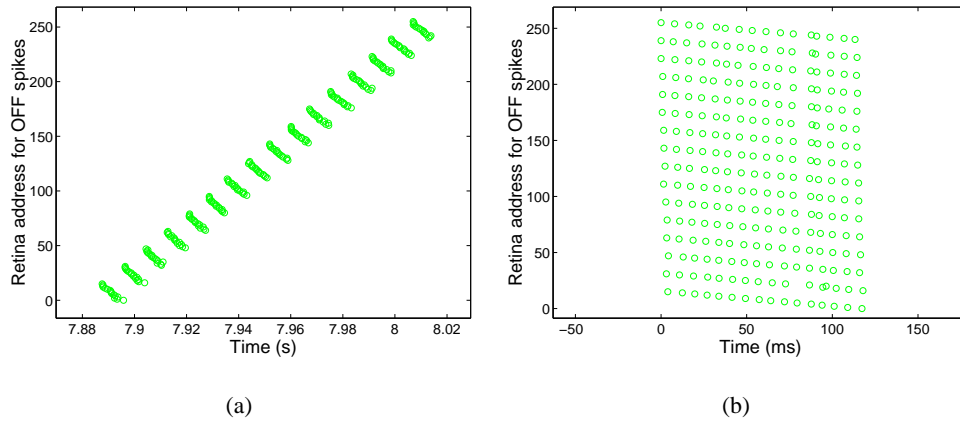

(a)                                                          (b)

Figure 5: The spike addresses from the retina were recorded when a 0 deg (a) and a 90 deg (b) oriented stimulus moved across the retina. The figure shows the time progression of the stimulated pixels (OFF spikes are marked with circles) as the 0 deg oriented stimulus (see Figure 3 for the orientation definition) passed over each row in (a). The address on the ordinate is defined as $16Y + X$. A similar observation is true of (b) for the ordering of the OFF-transient spikes when each column on the retina was stimulated by the 90 deg oriented stimulus.

$V_{\mathrm{tau}}$. By increasing $V_{\mathrm{tau}}$, we decrease the time constant of the neuron. Because the neuron charges up to threshold through the summation of the incoming EPSPs, it can only spike if the ISIs of the incoming spikes are small enough. The synaptic weight determines the number of EPSPs needed to drive the neuron above threshold.

We first investigated the feedforward model by using a template size of $5 \times 7$ (3 deg $\times$ 4.2 deg) for one neuron and $7 \times 5$ (4.2 deg $\times$ 3 deg) for the second neuron. The aspect ratio of this template was 1.4. (We have repeated the following experiments using smaller template sizes ($3 \times 5$ and $1 \times 3$) and the experimental results were pretty much the same.) The time constant of the neuron and synaptic gain and strength were adjusted so that both neurons responded optimally to the stimulus. The connection from the global inhibitory neuron to the two excitatory neurons was disabled.

Data was collected from the multi-neuron chip for different orientations of the drum (and hence of the stimulus). The stimulus was presented approximately 500–1000 times to the retina. Since the orientation-selective neurons responded with only 1–3 spikes every time the stimulus moved over the retina, we normalized the total number of spikes collected in these experiments to the number of stimulus presentations. The results are shown as a polar plot in Figure 6(a) for the two neurons that are sensitive to orthogonal orientations. Each neuron was more sensitive to a stimulus at its preferred orientation than the non-preferred orientations. The neuron responded more to the orthogonal orientation than to the in-between orientations because there were a small number of retina spikes that arrived with a small ISI when the orthogonally-oriented stimulus moved across the template space of the retina (see Figure 3). We used an orientation-selective (OS) index to quantify the orientation selectivity of the neuron. This index is defined as $\mathrm{OS} = \frac{\mathrm{R(preferred) - R(nonpreferred)}}{\mathrm{R(preferred) + R(nonpreferred)}}$ where R() is the response of the neuron. As an example, R(preferred) for neuron 5, which is sensitive to vertical orientations, is R(90)+R(270) and R(nonpreferred) is R(0)+R(180).

We next investigated the feedback model. In the presence of global inhibition, the multi-

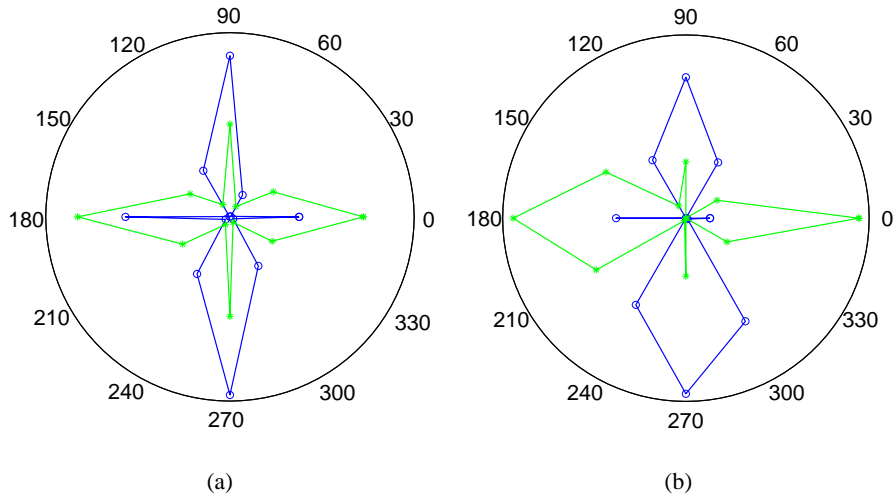

(a)                                              (b)

Figure 6: Orientation tuning curves of the two neurons in the (a) absence and (b) presence of global inhibition. The responses of the neurons were measured by the number of spikes collected per stimulus presentation. The radius of the polar plot is normalized to the maximum response of both neurons. The data was collected for stimulus orientations spaced at 30 deg intervals. The neuron that responded preferably to a 90 deg oriented stimulus (solid curve) also had a small response to a stimulus at 0 deg orientation (OS=0.428). The same observation is true for the other neuron (dashed curve) (OS=0.195). In the presence of global inhibition, each neuron responded less to the non-preferred orientation due to the suppression from the other neuron (cross-orientation inhibition). The output firing rates were also lower in this case (approximately half of the firing rates in the absence of inhibition). The OS indices are 0.546 (solid curve) and 0.497 (dashed curve) respectively.

neuron system acts like a soft winner-take-all circuit. We tuned the coupling strengths between the excitatory neurons and the inhibitory neuron so that we obtained the optimal response to the same stimulus presentations as in the feedforward case. The new tuning curves are plotted in Figure 6(b). The non-preferred response of a neuron was suppressed by the other neuron through the recurrent inhibition (cross-orientation inhibition).

## 5  Conclusion

We demonstrated a programmable multi-chip VLSI system that can be used for exploring spike-based processing models. This system has advantages over computer neuronal models in that it is real-time and the computational time does not scale with the size of the neuronal network. The spiking neurons can be configured for different computational properties. Interchip and intrachip connectivity between neurons can be programmed using the AER protocol. In this work, we created receptive fields for orientation-tuned spiking neurons by mapping the transient spikes from a silicon retina onto the neurons using a microcontroller. We have not mapped onto all the neurons on the transceiver chip because the PIC microcontroller we used is not fast enough to create receptive fields for more neurons without distorting the ISI distribution of the incoming retina spikes.

We evaluated the responses of the orientation-tuned spiking neurons for different receptive field sizes and aspect ratios and also in the absence and presence of feedback inhibition. In

a feedforward model, the aVLSI spiking neurons show orientation selectivity similar to digital simulations of continuous-valued neurons. Adding inhibition increased the selectivity of the spiking neurons between orthogonal orientations.

We can extend the multi-chip VLSI system in this work to a more sophisticated system that supports multiple senders and multiple receivers. Such a system can be used, for example, to implement multi-scale cortical models. The success of the system in this work opens up the way for more elaborate spike-based emulations in the future.

## 6  Acknowledgements

We acknowledge T. Horiuchi for the original design of the transceiver chip and David Lawrence for the software driver development in this work. This work was supported in part by the Swiss National Foundation Research SPP grant and the Köbler Foundation.

## References

Ben-Yishai, R., Bar-Or, R. L., and Sompolinsky, H. (1995). Theory of orientation tuning in visual cortex. *P. Natl. Acad. Sci. USA*, 92(9):3844–3848.

Boahen, K. A. (1996). A retinomorphic vision system. *IEEE Micro*, 16(5):30–39.

Boahen, K. A., Andreou, A., Hinck, T., Kramer, J., and Whatley, A. (1997). Computation- and memory-based projective field processors. In Sejnowski, T., Koch, C., and Douglas, R., editors, *Telluride NSF workshop on neuromorphic engineering*, Telluride, CO.

Douglas, R., Koch, C., Mahowald, M., Martin, K., and Suarez, H. (1995). Recurrent excitation in neocortical circuits. *Science*, 269(5226):981–985.

Douglas, R. and Martin, K. (1991). A functional microcircuit for cat visual cortex. *J. Physiol.*, 440:735–769.

Hubel, D. and Wiesel, T. (1962). Receptive fields, binocular interaction and functional architecture. *J. of Physio.(Lond)*, 160:106–154.

Kramer, J. (2001). An integrated optical transient sensor. Submitted for publication.

Lazzaro, J., Wawrzynek, J., Mahowald, M., Sivilotti, M., and Gillespie, D. (1993). Silicon auditory processors as computer peripherals. *IEEE Transactions on Neural Networks*, 4(3):523–528.

Liu, S.-C., Kramer, J., Indiveri, G., Delbruck, T., Burg, T., and Douglas, R. (2001). Orientation-selective aVLSI spiking neurons. *Neural Networks*, 14(6/7):629–643. Special Issue on Spiking Neurons in Neuroscience and Technology.

Somers, D., Nelson, S., and Sur, M. (1995). An emergent model of orientation selectivity in cat visual cortex simple cells. *Journal of Neuroscience*, 15(8):5448–5465.

Whatley, A., Kramer, J., and Douglas, R. (1997). ON/OFF retina to silicon cortex. In Sejnowski, T., Koch, C., and Douglas, R., editors, *Telluride NSF workshop on neuromorphic engineering*, Telluride, CO.
